# Support Vector Machines Applied to Face Recognition

**P. Jonathon Phillips**

National Institute of Standards and Technology
Bldg 225 / Rm A216
Gaithersburg, MD 20899
Tel 301.975.5348; Fax 301.975.5287
jonathon@nist.gov

## Abstract

Face recognition is a $K$ class problem, where $K$ is the number of known individuals; and support vector machines (SVMs) are a binary classification method. By reformulating the face recognition problem and re-interpreting the output of the SVM classifier, we developed a SVM-based face recognition algorithm. The face recognition problem is formulated as a problem in *difference space*, which models dissimilarities between two facial images. In difference space we formulate face recognition as a two class problem. The classes are: dissimilarities between faces of the same person, and dissimilarities between faces of different people. By modifying the interpretation of the decision surface generated by SVM, we generated a similarity metric between faces that is learned from examples of differences between faces. The SVM-based algorithm is compared with a principal component analysis (PCA) based algorithm on a difficult set of images from the FERET database. Performance was measured for both verification and identification scenarios. The identification performance for SVM is 77-78% versus 54% for PCA. For verification, the equal error rate is 7% for SVM and 13% for PCA.

## 1 Introduction

Face recognition has developed into a major research area in pattern recognition and computer vision. Face recognition is different from classical pattern-recognition problems such as character recognition. In classical pattern recognition, there are relatively few classes, and many samples per class. With many samples per class, algorithms can classify samples not previously seen by *interpolating* among the training samples. On the other hand, in

face recognition, there are many individuals (classes), and only a few images (samples) per person, and algorithms must recognize faces by *extrapolating* from the training samples. In numerous applications there can be only one training sample (image) of each person.

Support vector machines (SVMs) are formulated to solve a classical two class pattern recognition problem. We adapt SVM to face recognition by modifying the interpretation of the output of a SVM classifier and devising a representation of facial images that is concordant with a two class problem. Traditional SVM returns a binary value, the class of the object. To train our SVM algorithm, we formulate the problem in a *difference space*, which explicitly captures the dissimilarities between two facial images. This is a departure from traditional *face space* or *view-based* approaches, which encodes each facial image as a separate view of a face.

In difference space, we are interested in the following two classes: the dissimilarities between images of the same individual, and dissimilarities between images of different people. These two classes are the input to a SVM algorithm. A SVM algorithm generates a decision surface separating the two classes. For face recognition, we re-interpret the decision surface to produce a similarity metric between two facial images. This allows us to construct face-recognition algorithms. The work of Moghaddam et al. [3] uses a Bayesian method in a difference space, but they do not derive a similarity distance from both positive and negative samples.

We demonstrate our SVM-based algorithm on both verification and identification applications. In identification, the algorithm is presented with an image of an unknown person. The algorithm reports its best estimate of the identity of an unknown person from a database of known individuals. In a more general response, the algorithm will report a list of the most similar individuals in the database. In verification (also referred to as authentication), the algorithm is presented with an image and a claimed identity of the person. The algorithm either accepts or rejects the claim. Or, the algorithm can return a confidence measure of the validity of the claim.

To provide a benchmark for comparison, we compared our algorithm with a principal component analysis (PCA) based algorithm. We report results on images from the FERET database of images, which is the de facto standard in the face recognition community. From our experience with the FERET database, we selected harder sets of images on which to test the algorithms. Thus, we avoided saturating performance of either algorithm and providing a robust comparison between the algorithms. To test the ability of our algorithm to generalize to new faces, we trained and tested the algorithms on separate sets of faces.

## 2  Background

In this section we will give a brief overview of SVM to present the notation used in this paper. For details of SVM see Vapnik [7], or for a tutorial see Burges [1]. SVM is a binary classification method that finds the optimal linear decision surface based on the concept of structural risk minimization. The decision surface is a weighted combination of elements of the training set. These elements are called support vectors and characterize the boundary between the two classes. The input to a SVM algorithm is a set $\{(x_i, y_i)\}$ of labeled training data, where $x_i$ is the data and $y_i = -1$ or $1$ is the label. The output of a SVM algorithm is a set of $N_S$ support vectors $s_i$, coefficient weights $\alpha_i$, class labels $y_i$ of the support vectors, and a constant term $b$. The linear decision surface is

$$\mathbf{w} \cdot \mathbf{z} + b = 0,$$

where

$$\mathbf{w} = \sum_{i=1}^{N_S} \alpha_i y_i s_i.$$

SVM can be extended to nonlinear decision surfaces by using a kernel $K(\cdot, \cdot)$ that satisfies Mercer's condition [1, 7]. The nonlinear decision surface is

$$\sum_{i=1}^{N_S} \alpha_i y_i K(\mathbf{s_i}, \mathbf{z}) + b = 0.$$

A facial image is represented as a vector $\mathbf{p} \in \Re^N$, where $\Re^N$ is referred to as *face space*. Face space can be the original pixel values vectorized or another feature space; for example, projecting the facial image on the eigenvectors generated by performing PCA on a training set of faces [6] (also referred to as *eigenfaces*).

We write $\mathbf{p_1} \sim \mathbf{p_2}$ if $\mathbf{p_1}$ and $\mathbf{p_2}$ are images of the same face, and $\mathbf{p_1} \not\sim \mathbf{p_2}$ if they are images of different faces. To avoid confusion we adopted the following terminology for identification and verification. The *gallery* is the set of images of known people and a *probe* is an unknown face that is presented to the system. In identification, the face in a probe is identified. In verification, a probe is the facial image presented to the system whose identity is to be verified. The set of unknown faces is call the *probe set*.

## 3  Verification as a two class problem

Verification is fundamentally a two class problem. A verification algorithm is presented with an image $\mathbf{p}$ and a claimed identity. Either the algorithm accepts or rejects the claim. A straightforward method for constructing a classifier for person $X$, is to feed a SVM algorithm a training set with one class consisting of facial images of person $X$ and the other class consisting of facial images of other people. A SVM algorithm will generated a linear decision surface, and the identity of the face in image $\mathbf{p}$ is accepted if

$$\mathbf{w} \cdot \mathbf{p} + b \leq 0,$$

otherwise the claim is rejected.

This classifier is designed to minimizes the structural risk. Structural risk is an overall measure of classifier performance. However, verification performance is usually measured by two statistics, the probability of correct verification, $P_V$, and the probability of false acceptance, $P_F$. There is a tradeoff between $P_V$ and $P_F$. At one extreme all claims are rejected and $P_V = P_F = 0$; and at the other extreme, all claims are accepted and $P_V = P_F = 1$. The operating values for $P_V$ and $P_F$ are dictated by the application.

Unfortunately, the decision surface generated by a SVM algorithm produces a single performance point for $P_V$ and $P_F$. To allow for adjusting $P_V$ and $P_F$, we parameterize a SVM decision surface by $\Delta$. The parametrized decision surface is

$$\mathbf{w} \cdot \mathbf{z} + b = \Delta,$$

and the identity of the face image $p$ is accepted if

$$\mathbf{w} \cdot \mathbf{p} + b \leq \Delta.$$

If $\Delta = -\infty$, then all claims are rejected and $P_V = P_F = 0$; if $\Delta = +\infty$, all claims are accepted and $P_V = P_F = 0$. By varying $\Delta$ between negative and positive infinity, all possible combinations of $P_V$ and $P_F$ are found.

Nonlinear parametrized decision surfaces are described by

$$\sum_{i=1}^{N_S} \alpha_i y_i K(\mathbf{s_i}, \mathbf{z}) + b = \Delta.$$

## 4  Representation

In a canonical face recognition algorithm, each individual is a class and the distribution of each face is estimated or approximated. In this method, for a gallery of $K$ individuals, the identification problem is a $K$ class problem, and the verification problem is $K$ instances of a two class problems. To reduce face recognition to a single instance of a two class problem, we introduce a new representation. We model the dissimilarities between faces. Let $T = \{t_1, \ldots, t_M\}$ be a training set of faces of $K$ individuals, with multiple images of each of the $K$ individuals. From $T$, we generate two classes. The first is the *within-class differences set*, which are the dissimilarities in facial images of the same person. Formally the within-class difference set is

$$C_1 = \{\mathbf{t_i} - \mathbf{t_j} | \mathbf{t_i} \sim \mathbf{t_j}\}.$$

The set $C_1$ contains within-class differences for all $K$ individuals in $T$, not dissimilarities for one of the $K$ individuals in the training set. The second is the *between-class differences set*, which are the dissimilarities among images of different individuals in the training set. Formally,

$$C_2 = \{\mathbf{t_i} - \mathbf{t_j} | \mathbf{t_i} \not\sim \mathbf{t_j}\}.$$

Classes $C_1$ and $C_2$ are the inputs to our SVM algorithm, which generates a decision surface. In the pure SVM paradigm, given the difference between facial images $\mathbf{p_1}$ and $\mathbf{p_2}$, the classifier estimates if the faces in the two images are from the same person. In the modification described in section 3, the classification returns a measure of similarity $\delta = \mathbf{w} \cdot (\mathbf{p_1} - \mathbf{p_2}) + b$. This similarity measure is the basis for the SVM-based verification and identification algorithms presented in this paper.

## 5  Verification

In verification, there is a gallery $\{g_j\}$ of $m$ known individuals. The algorithm is presented with a probe $p$ and a claim to be person $j$ in the gallery. The first step of the verification algorithm computes the similarity score

$$\delta = \sum_{i=1}^{N_S} \alpha_i y_i K(\mathbf{s_i}, \mathbf{g_j} - \mathbf{p}) + b.$$

The second step accepts the claim if $\delta \leq \Delta$. Otherwise, the claim is rejected. The value of $\Delta$ is set to meet the desired tradeoff between $P_V$ and $P_F$.

## 6  Identification

In identification, there is a gallery $\{g_j\}$ of $m$ known individuals. The algorithm is presented with a probe $p$ to be identified. The first step of the identification algorithm computes a similarity score between the probe and each of the gallery images. The similar score between $p$ and $g_j$ is

$$\delta_j = \sum_{i=1}^{N_S} \alpha_i y_i K(\mathbf{s_i}, \mathbf{g_j} - \mathbf{p}) + b.$$

In the second step, the probe is identified as person $j$ that has minimum similarity score $\delta_j$. An alternative method of reporting identification results is to order the gallery by the similarity measure $\delta_j$.

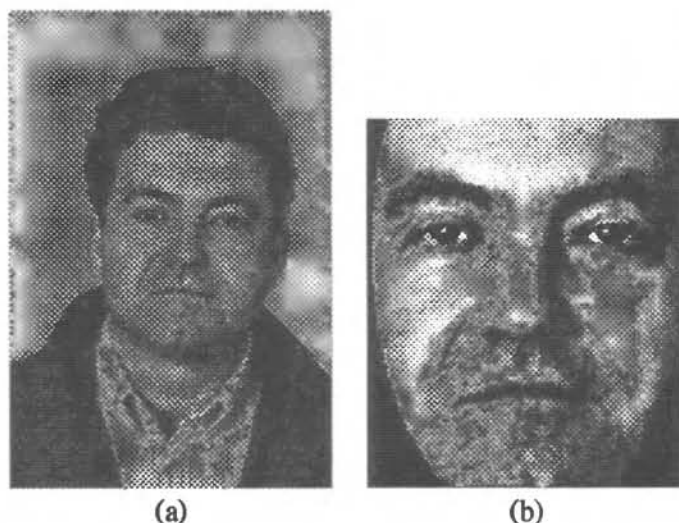

Figure 1: (a) Original image from the FERET database. (b) Image after preprocessing.

## 7 Experiments

We demonstrate our SVM-based verification and identification algorithms on 400 frontal images from the FERET database of facial images [5]. To provide a benchmark for algorithm performance, we provide performance for a PCA-based algorithm on the same set of images. The PCA algorithm identifies faces with a $L_2$ nearest neighbor classifier. For the SVM-based algorithms, a radial basis kernel was used.

The 400 images consisted of two images of 200 individuals, and were divided into disjoint training and testing sets. Each set consisted of two images of 100 people. All 400 images were preprocessed to normalize geometry and illumination, and to remove background and hair (figure 1). The preprocessing procedure consisted of manually locating the centers of the eyes; translating, rotating, and scaling the faces to place the center of the eyes on specific pixels; masking the faces to remove background and hair; histogram equalizing the non-masked facial pixels; and scaling the non-masked facial pixels to have zero mean and unit variance.

PCA was performed on 100 preprocessed images (one image of each person in the training set). This produced 99 eigenvectors $\{e_\ell\}$ and eigenvalues $\{\lambda_\ell\}$. The eigenvectors were ordered so that $\lambda_i < \lambda_j$ when $i < j$. Thus, the low order eigenvectors encode the majority of the variance in the training set. The faces were represented by projecting them on a subset of the eigenvectors and this is the face space. We varied the dimension of face space by changing the number of eigenvectors in the representation.

In all experiments, the SVM training set consisted of the same images. The SVM-training set $T$ consisted of two images of 50 individuals from the general training set of 100 individuals. The set $C_1$ consisted of all 50 within-class differences from faces of the same individuals. The set $C_2$ consisted of 50 randomly selected between-class differences.

The verification and identification algorithms were tested on a gallery consisted of 100 images from the test set, with one image person. The probe set consisted of the remaining images in the test set (100 individuals, with one image per person).

We report results for verification on a face space that consisted of the first 30 eigenfeatures (an eigenfeature is the projection of the image onto an eigenvector). The results are reported as a receiver operator curve (ROC) in figure 2. The ROC in figure 2 was computed

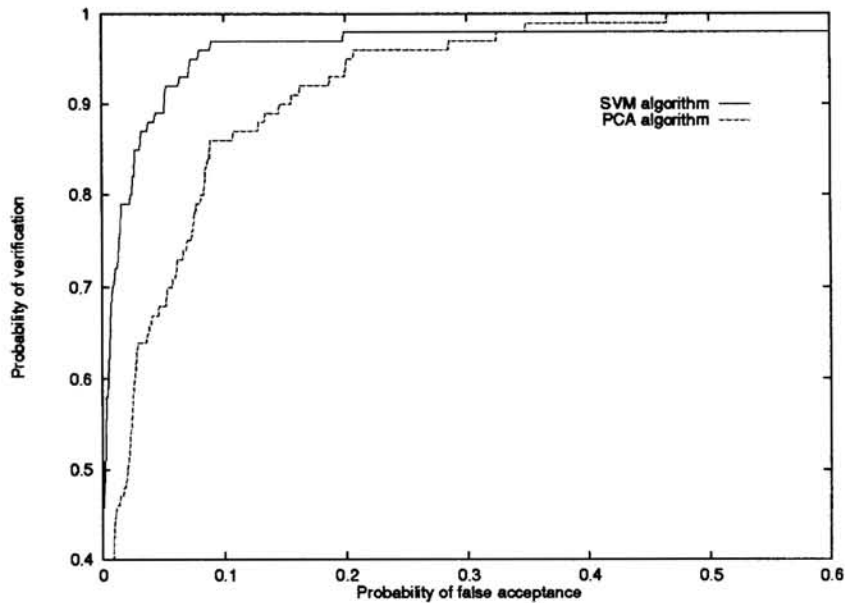

Figure 2: ROC for verification (using first 30 eigenfeatures).

by averaging the ROC for each of the 100 individuals in the gallery. For person $g_j$, the probe set consisted of one image of person $g_j$ and 99 faces of different people. A summary statistic for verification is the equal error rate. The equal error rate is the point where the probability of false acceptance is equal to the probability of false verification, or mathematically, $P_F = 1 - P_V$. For the SVM-based algorithm the equal error rate is 0.07, and for the PCA-based algorithm is 0.13.

For identification, the algorithm estimated the identity of each of the probes in the probe set. We compute the probability of correctly identifying the probes for a set of face spaces parametrized by the number of eigenfeatures. We always use the first $n$ eigenfeatures, thus we are slowly increasing the amount of information, as measured by variance, available to the classifier. Figure 3 shows probability of identification as a function of representing faces by the first $n$ eigenfeatures. PCA achieves a correct identification rate of 54% and SVM achieves an identification rate of 77-78%. (The PCA results we report are significantly lower than those reported in the literature [2, 3]. This is because we selected a set of images that are more difficult to recognize. The results are consistent with experimentations in our group with PCA-based algorithms on the FERET database [4]. We selected this set of images so that performance of neither the PCA or SVM algorithms are saturated.)

## 8   Conclusion

We introduced a new technique for applying SVM to face recognition. We demonstrated the algorithm on both verification and identification applications. We compared the performance of our algorithm to a PCA-based algorithm. For verification, the equal error rate of our algorithm was almost half that of the PCA algorithm, 7% versus 13%. For identification, the error of SVM was half that of PCA, 22-23% versus 46%. This indicates that SVM is making more efficient use of the information in face space than the baseline PCA algorithm.

One of the major concerns in practical face recognition applications is the ability of the

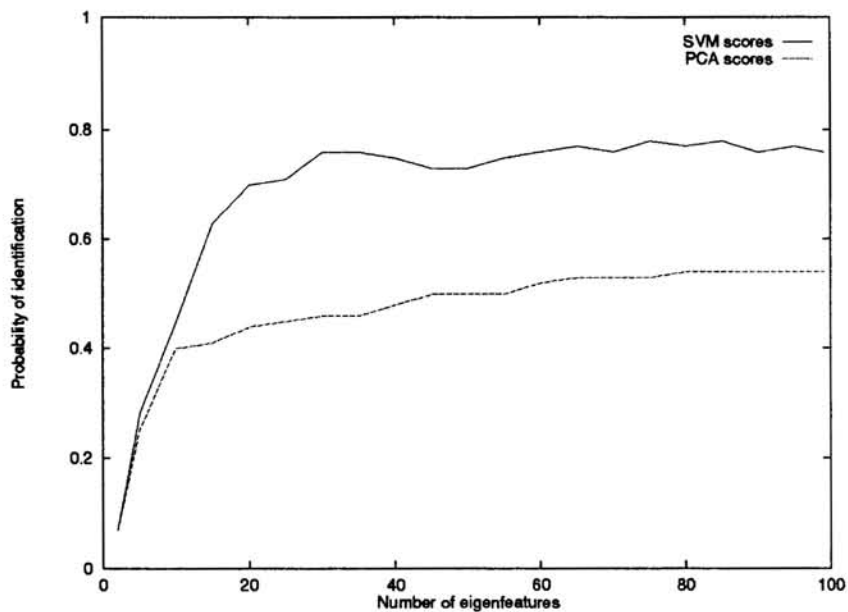

Figure 3: Probability of identification as a function of the number eigenfeatures.

algorithm to generalize from a training set of faces to faces outside of the training set. We demonstrated the ability of the SVM-based algorithm to generalize by training and testing on separate sets.

Future research directions include varying the kernel $K$, changing the representation space, and expanding the size of the gallery and probe set. There is nothing in our method that is specific to faces, and it should generalize to other biometrics such as fingerprints.

# References

[1] C. J. C. Burges. A tutorial on support vector machines for pattern recognition. *Data mining and knowledge discovery*, (submitted), 1998.

[2] B. Moghaddam and A. Pentland. Face recognition using view-based and modular eigenspaces. In *Proc. SPIE Conference on Automatic Systems for the Identification and Inspection of Humans*, volume SPIE Vol. 2277, pages 12–21, 1994.

[3] B. Moghaddam, W. Wahid, and A. Pentland. Beyond eigenfaces: probablistic matching for face recognition. In *3rd International Conference on Automatic Face and Gesture Recognition*, pages 30–35, 1998.

[4] H. Moon and P. J. Phillips. Analysis of PCA-based face recognition algorithms. In K. W. Bowyer and P. J. Phillips, editors, *Empirical Evaluation Techniques in Computer Vision*. IEEE Computer Society Press, Los Alamitos, CA, 1998.

[5] P. J. Phillips, H. Wechsler, J. Huang, and P. Rauss. The FERET database and evaluation procedure for face-recognition algorithms. *Image and Vision Computing Journal*, 16(5):295–306, 1998.

[6] M. Turk and A. Pentland. Eigenfaces for recognition. *J. Cognitive Neuroscience*, 3(1):71–86, 1991.

[7] V. Vapnik. *The nature of statistical learning theory*. Springer, New York, 1995.
